# Computation of Smooth Optical Flow in a Feedback Connected Analog Network

**Alan Stocker** *
Institute of Neuroinformatics
University and ETH Zürich
Winterthurerstrasse 190
8057 Zürich, Switzerland

**Rodney Douglas**
Institute of Neuroinformatics
University and ETH Zürich
Winterthurerstrasse 190
8057 Zürich, Switzerland

## Abstract

In 1986, Tanner and Mead [1] implemented an interesting constraint satisfaction circuit for global motion sensing in aVLSI. We report here a new and improved aVLSI implementation that provides smooth optical flow as well as global motion in a two dimensional visual field. The computation of optical flow is an ill-posed problem, which expresses itself as the aperture problem. However, the optical flow can be estimated by the use of regularization methods, in which additional constraints are introduced in terms of a global energy functional that must be minimized. We show how the algorithmic constraints of Horn and Schunck [2] on computing smooth optical flow can be mapped onto the physical constraints of an equivalent electronic network.

## 1 Motivation

The perception of apparent motion is crucial for navigation. Knowledge of local motion of the environment relative to the observer simplifies the calculation of important tasks such as time-to-contact or focus-of-expansion. There are several methods to compute optical flow. They have the common problem that their computational load is large. This is a severe disadvantage for autonomous agents, whose computational power is restricted by energy, size and weight. Here we show how the global regularization approach which is necessary to solve for the ill-posed nature of computing optical flow, can be formulated as a local feedback constraint, and implemented as a physical analog device that is computationally efficient.

## 2   Smooth Optical Flow

Horn and Schunck [2] defined optical flow in relation to the spatial and temporal changes in image brightness. Their model assumes that the total image brightness $E(x, y, t)$ does not change over time;

$$\frac{d}{dt}E(x, y, t) = 0. \tag{1}$$

Expanding equation (1) according to the chain rule of differentiation leads to

$$F \equiv \frac{\delta}{\delta x}E(x, y, t)u + \frac{\delta}{\delta y}E(x, y, t)v + \frac{\delta}{\delta t}E(x, y, t) = 0, \tag{2}$$

where $u = dx/dt$ and $v = dy/dt$ represent the two components of the local optical flow vector.

Since there is one equation for two unknowns at each spatial location, the problem is ill-posed, and there are an infinite number of possible solutions lying on the *constraint line* for every location $(x, y)$. However, by introducing an additional constraint the problem can be regularized and a unique solution can be found.

For example, Horn and Schunck require the optical flow field to be smooth. As a measure of smoothness they choose the squares of of the spatial derivatives of the flow vectors,

$$S^2 = \left(\frac{\delta u}{\delta x}\right)^2 + \left(\frac{\delta u}{\delta y}\right)^2 + \left(\frac{\delta v}{\delta x}\right)^2 + \left(\frac{\delta v}{\delta y}\right)^2. \tag{3}$$

One can also view this constraint as introducing *a priori* knowledge: the closer two points are in the image space the more likely they belong to the projection of the same object. Under the assumption of rigid objects undergoing translational motion, this constraint implies that the points have the same, or at least very similar motion vectors. This assumption is obviously not valid at boundaries of moving objects, and so this algorithm fails to detect motion discontinuities [3].

The computation of smooth optical flow can now be formulated as the minimization problem of a global energy functional,

$$\iint \underbrace{F^2 + \lambda S^2}_{L} \, dx \, dy \; \longrightarrow \; min \tag{4}$$

with $F$ and $S^2$ as in equation (2) and (3) respectively. Thus, we exactly apply the approach of *standard regularization theory* [4]:

$$\begin{aligned}
&\mathbf{Ax} = \mathbf{y} &&\text{y: data} \\
&\mathbf{x} = \mathbf{A}^{-1}\mathbf{y} &&\text{inverse problem, ill-posed} \\
&\| \, \mathbf{Ax} - \mathbf{y} \, \| + \lambda \| \, \mathbf{P} \, \| = min &&\text{regularization}
\end{aligned}$$

The regularization parameter, $\lambda$, controls the degree of smoothing of the solution and its closeness to the data. The norm, $\| \cdot \|$, is quadratic. A difference in our case is that $\mathbf{A}$ is not constant but depends on the data. However, if we consider motion on a discrete time-axis and look at snapshots rather than continuously changing images, $\mathbf{A}$ is *quasi-stationary*.[1] The energy functional (4) is convex and so, a simple numerical technique like gradient descent would be able to find the global minimum. To compute optical flow while preserving motion discontinuities one can modify the energy functional to include a binary line process that prevents smoothing over discontinuities [4]. However, such an functional will not be convex. Gradient descent methods would probably fail to find the global amongst all local minima and other methods have to be applied.

## 3   A Physical Analog Model

### 3.1   Continuous space

Standard regularization problems can be mapped onto electronic networks consisting of conductances and capacitors [5]. Hutchinson et al. [6] showed how resistive networks can be used to compute optical flow and Poggio et al. [7] introduced electronic network solutions for second-order-derivative optic flow computation. However, these proposed network architectures all require complicated and sometimes negative conductances although Harris et al. [8] outlined a similar approach as proposed in this paper independently. Furthermore, such networks were not implemented practically, whereas our implementation with constant nearest neighbor conductances is intuitive and straightforward.
Consider equation (4):

$$L = L(u, v, \nabla u, \nabla v, x, y).$$

The *Lagrange function L* is sufficiently regular ($L \in C^2$), and thus it follows from calculus of variation that the solution of equation (4) also suffices the linear Euler-Lagrange equations

$$\lambda \nabla^2 u - E_x(E_x u + E_y v + E_t) = 0 \qquad (5)$$
$$\lambda \nabla^2 v - E_y(E_x u + E_y v + E_t) = 0.$$

The Euler-Lagrange equations are only necessary conditions for equation (4). The sufficient condition for solutions of equations (5) to be a weak minimum is the strong Legendre-condition, that is

$$L_{\nabla u \nabla u} > 0 \qquad \text{and} \qquad L_{\nabla v \nabla v} > 0,$$

which is easily shown to be true.

### 3.2   Discrete Space – Mapping to Resistive Network

By using a discrete five-point approximation of the Laplacian $\nabla^2$ on a regular grid, equations (5) can be rewritten as

$$\lambda(u_{i+1,j} + u_{i-1,j} + u_{i,j+1} + u_{i,j-1} - 4u_{i,j}) - E_{x_{i,j}}(E_{x_{i,j}} u_{i,j} + E_{y_{i,j}} v_{i,j} + E_{t_{i,j}}) = 0 \quad (6)$$
$$\lambda(v_{i+1,j} + v_{i-1,j} + v_{i,j+1} + v_{i,j-1} - 4v_{i,j}) - E_{y_{i,j}}(E_{x_{i,j}} u_{i,j} + E_{y_{i,j}} v_{i,j} + E_{t_{i,j}}) = 0$$

where $i$ and $j$ are the indices for the sampling nodes. Consider a single node of the resistive network shown in Figure 1:

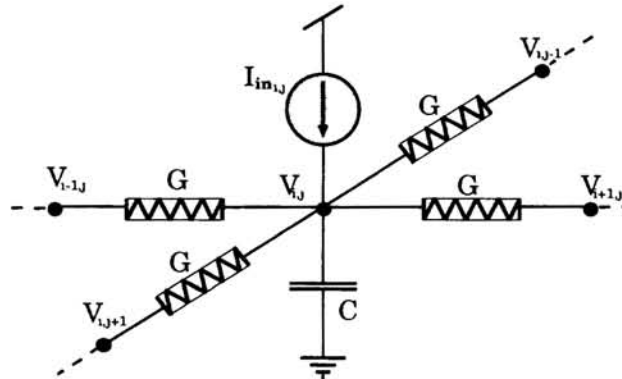

Figure 1: Single node of a resistive network.

From Kirchhoff's law it follows that

$$C \frac{dV_{i,j}}{dt} = G(V_{i+1,j} + V_{i-1,j} + V_{i,j+1} + V_{i,j-1} - 4V_{i,j}) + I_{in_{i,j}} \qquad (7)$$

where $V_{i,j}$ represents the voltage and $I_{in_{i,j}}$ the input current. $G$ is the conductance between two neighboring nodes and $C$ the node capacitance.

In steady state, equation (7) becomes

$$G(V_{i+1,j} + V_{i-1,j} + V_{i,j+1} + V_{i,j-1} - 4V_{i,j}) + I_{in_{i,j}} = 0. \qquad (8)$$

The analogy with equations (6) is obvious:

$$
\begin{aligned}
G &\longleftrightarrow \lambda \\
Iu_{in_{i,j}} &\longleftrightarrow -E_{x_{i,j}}(E_{x_{i,j}}u_{i,j} + E_{y_{i,j}}v_{i,j} + E_{t_{i,j}}) \\
Iv_{in_{i,j}} &\longleftrightarrow -E_{y_{i,j}}(E_{x_{i,j}}u_{i,j} + E_{y_{i,j}}v_{i,j} + E_{t_{i,j}})
\end{aligned} \qquad (9)
$$

To create the full system we use two parallel resistive networks in which the node voltages $U_{i,j}$ and $V_{i,j}$ represent the two components of the optical flow vector $u$ and $v$. The input currents $Iu_{in_{i,j}}$ and $Iv_{in_{i,j}}$ are computed by a negative *recurrent feedback loop* modulated by the input data, which are the spatial and temporal intensity gradients.

Notice that the input currents are proportional to the deviation of the *local brightness constraint*: the less the local optical flow solution fits the data the higher the current $I_{in_{i,j}}$ will be to correct the solution and vice versa.

Stability and convergence of the network are guaranteed by Maxwell's minimum power principle [4, 9].

## 4    The Smooth Optical Flow Chip

### 4.1    Implementation

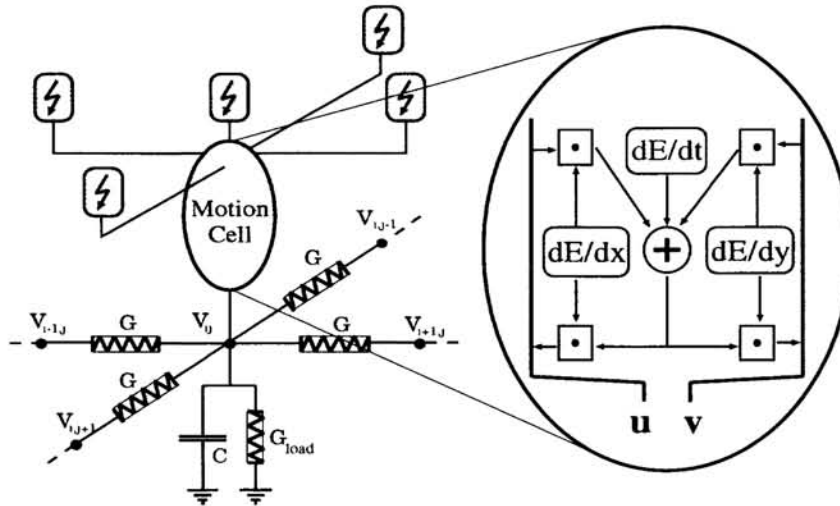

Figure 2: A single motion cell within the three layer network. For simplicity only one resistive network is shown.

The circuitry consists of three functional layers (Figure 2). The input layer includes an array of adaptive photoreceptors [10] and provides the derivatives of the image brightness to the second layer. The spatial gradients are the first-order linear approximation obtained by subtracting the two neighboring photoreceptor outputs. The second layer computes the input current to the third layer according to equations (9). Finally these currents are fed into the two resistive networks that report the optical flow components.

The schematics of the core of a single motion cell are drawn in Figure 3. The photoreceptor and the temporal differentiator are not shown as well as the other half of the circuitry that computes the y-component of the flow vector.

A few remarks are appropriate here: First, the two components of the optical flow vector have to be able to take on positive and negative values with respect to some reference potential. Therefore, a symmetrical circuit scheme is applied where the positive and negative (reference voltage) values are carried on separate signal lines. Thus, the actual value is encoded as the difference of the two potentials.

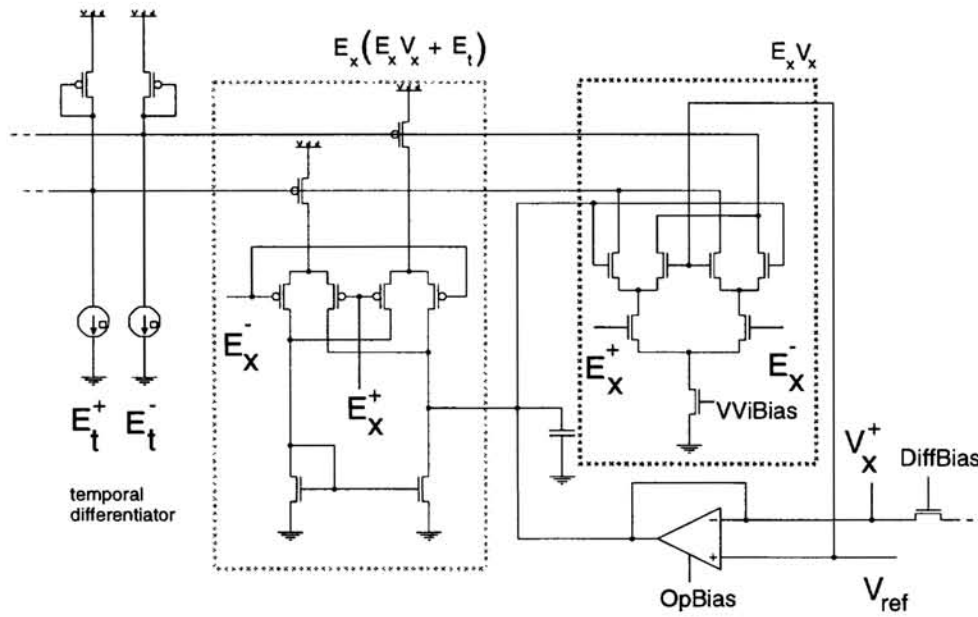

Figure 3: Cell core schematics; only the circuitry related to the computation of the x-component of the flow vector is shown.

Second, the limited linear range of the Gilbert multipliers leads to a narrow span of flow velocities that can be computed reliably. However, the tuning can be such that the operational range is either at high or very low velocities. Newer implementations are using modified multipliers with a larger linear range.

Third, consider a single motion cell (Figure 2). In principle, this cell would be able to satisfy the local constraint perfectly. In practice (see Figure 3), the finite output impedance of the p-type Gilbert multiplier slightly degrades this ideal solution by imposing an effective conductance $G_{load}$. Thus, a constant voltage on the capacitor representing a non-zero motion signal requires a net output current of the multiplier to maintain it. This requirement has two interesting consequences:

i) The reported optical flow is dependent on the spatial gradients (contrast). A single uncoupled cell according to Figure 2 has a steady state solution with

$$U_{i,j} \sim \frac{-E_{t_{i,j}} E_{x_{i,j}}}{(G_{load} + E_{x_{i,j}}^2 + E_{y_{i,j}}^2)} \quad \text{and} \quad V_{i,j} \sim \frac{-E_{t_{i,j}} E_{y_{i,j}}}{(G_{load} + E_{x_{i,j}}^2 + E_{y_{i,j}}^2)}$$

respectively. For the same object speed, the chip reports higher velocity signals for higher spatial gradients. Preferably, $G_{load}$ should be as low as possible to minimize its influence on the solution.

ii) On the other hand, the locally ill-posed problem is now well-posed because $G_{load}$ imposes a second constraint. Thus, the chip behaves sensibly in the case of low contrast input (small gradients), reporting zero motion where otherwise, unreliable high values would occur. This is convenient because the signal-to-noise ratio at low contrast is very poor. Furthermore, a single cell is forced to report the velocity on the constraint line with smallest absolute value, which is normal to the spatial gradient. That means that the chip

reports *normal flow* when there is no neighbor connection. Since there is an trade-off between the robustness of the optical flow computation and a low conductance $G_{load}$, the follower-connected transconductance amplifier in our implementation allows us to control $G_{load}$ above its small intrinsic value.

## 4.2 Results

The results reported below were obtained from a MOSIS tinychip containing a 7x7 array of motion cells each 325x325 $\lambda^2$ in size. The chip was fabricated in 1.2 $\mu m$ technology at AMI.

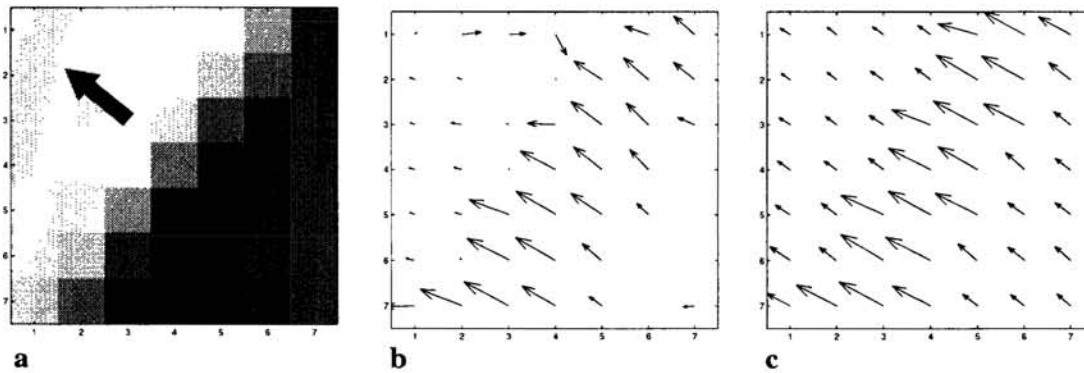

Figure 4: Smooth optical flow response of the chip to an left-upwards moving edge. **a**: photoreceptor output, the arrow indicates the actual motion direction. **b**: weak coupling (small conductance $G$). **c**: strong coupling.

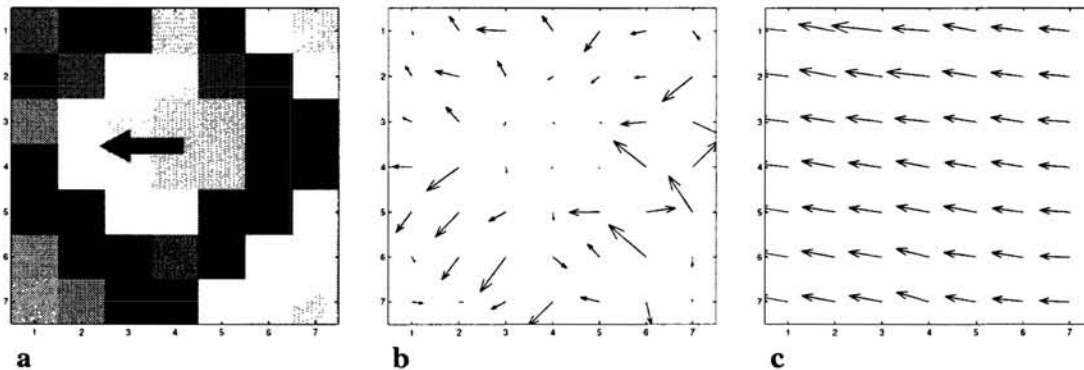

Figure 5: Response of the optical flow chip to a plaid stimulus moving towards the left: **a**: photoreceptor output; **b** shows the normal flow computation with disabled coupling between the motion cells in the network while in **c** the coupling strength is at maximum.

The chip is able to compute smooth optical flow in a qualitative manner. The smoothness can be set by adjusting the coupling conductances (Figure 4). Figure 5b presents the normal flow computation that occurs when the coupling between the motion cells is disabled. The limited resolution of this prototype chip together with the small size of the stimulus leads to a noisy response. However it is clear that the chip perceives the two gratings as separate moving objects with motion normal to their edge orientation. When the network

conductance is set very high the chip performs a collective computation solving the aperture problem under the assumption of single object motion. Figure 5c shows how the chip can compute the correct motion of a plaid pattern.

## 5   Conclusion

We have presented here an aVLSI implementation of a network that computes 2D smooth optical flow. The strength of the resistive coupling can be varied continuously to obtain different degrees of smoothing, from a purely local up to a single global motion signal. The chip ideally computes smooth optical flow in the classical definition of Horn and Schunck. Instead of using negative and complex conductances we implemented a network solution where each motion cell is performing a local constraint satisfaction task in a recurrent negative feedback loop.

It is significant that the solution of a global energy minimization task can be achieved within a network of local constraint solving cells that do not have explicit access to the global computational goal.

### Acknowledgments

This article is dedicated to Misha Mahowald. We would like to thank Eric Vittoz, Jörg Kramer, Giacomo Indiveri and Tobi Delbrück for fruitful discussions. We thank the Swiss National Foundation for supporting this work and MOSIS for chip fabrication.

## Footnotes

\* correspondence to: alan@ini.phys.ethz.ch

[1] In the aVLSI implementation this requires a much shorter settling time constant for the network than the brightness changes in the image.

## References

[1] J. Tanner and C.A. Mead. An integrated analog optical motion sensor. In S.-Y. Kung, R. Owen, and G. Nash, editors, *VLSI Signal Processing, 2*, page 59 ff. IEEE Press, 1986.

[2] B.K. Horn and B.G. Schunck. Determining optical flow. *Artificial Intelligence*, 17:185–203, 1981.

[3] A. Yuille. Energy functions for early vision and analog networks. *Biological Cybernetics*, 61:115–123, 1989.

[4] T. Poggio, V. Torre, and C. Koch. Computational vision and regularization theory. *Nature*, 317(26):314–319, September 1985.

[5] B. K. Horn. Parallel networks for machine vision. Technical Report 1071, MIT AI Lab, December 1988.

[6] J. Hutchinson, C. Koch, J. Luo, and C. Mead. Computing motion using analog and binary resistive networks. *Computer*, 21:52–64, March 1988.

[7] T. Poggio, W. Yang, and V. Torre. Optical flow: Computational properties and networks, biological and analog. *The Computing Neuron*, pages 355–370, 1989.

[8] J.G. Harris, C. Koch, E. Staats, and J. Luo. Analog hardware for detecting discontinuities in early vision. *Int. Journal of Computer Vision*, 4:211–223, 1990.

[9] J. Wyatt. Little-known properties of resistive grids that are useful in analog vision chip designs. In C. Koch and H. Li, editors, *Vision Chips: Implementing Vision Algorithms with Analog VLSI Circuits*, pages 72–89. IEEE Computer Society Press, 1995.

[10] S.C. Liu. Silicon retina with adaptive filtering properties. In *Advances in Neural Information Processing Systems 10*, November 1997.